# Analysis of Linsker's Simulations
# of Hebbian rules

**David J. C. MacKay**
Computation and Neural Systems
Caltech 164–30 CNS
Pasadena, CA 91125
mackay@aurel.cns.caltech.edu

**Kenneth D. Miller**
Department of Physiology
University of California
San Francisco, CA 94143 - 0444
ken@phyb.ucsf.edu

## ABSTRACT

Linsker has reported the development of centre–surround receptive fields and oriented receptive fields in simulations of a Hebb–type equation in a linear network. The dynamics of the learning rule are analysed in terms of the eigenvectors of the covariance matrix of cell activities. Analytic and computational results for Linsker's covariance matrices, and some general theorems, lead to an explanation of the emergence of centre–surround and certain oriented structures.

Linsker [Linsker, 1986, Linsker, 1988] has studied by simulation the evolution of weight vectors under a Hebb–type teacherless learning rule in a feed–forward linear network. The equation for the evolution of the weight vector **w** of a single neuron, derived by ensemble averaging the Hebbian rule over the statistics of the input patterns, is:[1]

$$\frac{\partial}{\partial t} w_i = k_1 + \sum_j (Q_{ij} + k_2) w_j \quad \text{subject to} \quad -w_{\max} \le w_i \le w_{\max} \qquad (1)$$

where $\mathbf{Q}$ is the covariance matrix of activities of the inputs to the neuron. The covariance matrix depends on the covariance function, which describes the dependence of the covariance of two input cells' activities on their separation in the input field, and on the location of the synapses, which is determined by a synaptic density function. Linsker used a gaussian synaptic density function.

Depending on the covariance function and the two parameters $k_1$ and $k_2$, different weight structures emerge. Using a gaussian covariance function (his layer $\mathcal{B} \to \mathcal{C}$), Linsker reported the emergence of non–trivial weight structures, ranging from saturated structures through centre–surround structures to bi–lobed oriented structures.

The analysis in this paper examines the properties of equation (1). We concentrate on the gaussian covariances in Linsker's layer $\mathcal{B} \to \mathcal{C}$, and give an explanation of the structures reported by Linsker. Several of the results are more general, applying to any covariance matrix $\mathbf{Q}$. Space constrains us to postpone general discussion, and criteria for the emergence of centre–surround weight structures, technical details, and discussion of other model networks, to future publications [MacKay, Miller, 1990].

# 1   ANALYSIS IN TERMS OF EIGENVECTORS

We write equation (1) as a first order differential equation for the weight vector $\mathbf{w}$:

$$\dot{\mathbf{w}} = (\mathbf{Q} + k_2\mathbf{J})\mathbf{w} + k_1\mathbf{n} \quad \text{subject to} \quad -w_{\max} \leq w_i \leq w_{\max} \qquad (2)$$

where $\mathbf{J}$ is the matrix $J_{ij} = 1 \; \forall i, j$, and $\mathbf{n}$ is the DC vector $n_i = 1 \; \forall i$. This equation is linear, up to the hard limits on $w_i$. These hard limits define a hypercube in weight space within which the dynamics are confined. We make the following assumption:

**Assumption 1** *The principal features of the dynamics are established before the hard limits are reached. When the hypercube is reached, it captures and preserves the existing weight structure with little subsequent change.*

The matrix $\mathbf{Q}+k_2\mathbf{J}$ is symmetric, so it has a complete orthonormal set of eigenvectors[2] $\mathbf{e}^{(a)}$ with real eigenvalues $\lambda_a$. The linear dynamics within the hypercube can be characterised in terms of these eigenvectors, each of which represents an independently evolving weight configuration. First, equation (2) has a fixed point at

$$\mathbf{w}^{\mathrm{FP}} = -k_1(\mathbf{Q} + k_2\mathbf{J})^{-1}\mathbf{n} = -k_1 \sum_a \frac{\mathbf{e}^{(a)} \cdot \mathbf{n}}{\lambda_a}\mathbf{e}^{(a)} \qquad (3)$$

Second, relative to the fixed point, the component of $\mathbf{w}$ in the direction of an eigenvector grows or decays exponentially at a rate proportional to the corresponding eigenvalue. Writing $\mathbf{w}(t) = \sum_a w_a(t)\mathbf{e}^{(a)}$, equation (2) yields

$$w_a(t) - w_a^{\mathrm{FP}} = (w_a(0) - w_a^{\mathrm{FP}})e^{\lambda_a t} \qquad (4)$$

Thus, the principal emergent features of the dynamics are determined by the following three factors:

1. The principal eigenvectors of $\mathbf{Q} + k_2\mathbf{J}$, that is, the eigenvectors with largest positive eigenvalues. These are the fastest growing weight configurations.

2. Eigenvectors of $\mathbf{Q} + k_2\mathbf{J}$ with negative eigenvalue. Each is associated with an attracting constraint surface, the hyperplane defined by $w_a = w_a^{\text{FP}}$.

3. The location of the fixed point of equation (1). This is important for two reasons: a) it determines the location of the constraint surfaces; b) the fixed point gives a "head start" to the growth rate of eigenvectors $e^{(a)}$ for which $|w_a^{\text{FP}}|$ is large compared to $|w_a(0)|$.

## 2    EIGENVECTORS OF Q

We first examine the eigenvectors and eigenvalues of $\mathbf{Q}$. The principal eigenvector of $\mathbf{Q}$ dominates the dynamics of equation (2) for $k_1 = 0$, $k_2 = 0$. The subsequent eigenvectors of $\mathbf{Q}$ become important as $k_1$ and $k_2$ are varied.

### 2.1    PROPERTIES OF CIRCULARLY SYMMETRIC SYSTEMS

If an operator commutes with the rotation operator, its eigenfunctions can be written as eigenfunctions of the rotation operator. For Linsker's system, in the continuum limit, the operator $\mathbf{Q} + k_2\mathbf{J}$ is unchanged under rotation of the system. So the eigenfunctions of $\mathbf{Q} + k_2\mathbf{J}$ can be written as the product of a radial function and one of the angular functions $\cos l\theta$, $\sin l\theta$, $l = 0, 1, 2...$ To describe these eigenfunctions we borrow from quantum mechanics the notation $n = 1, 2, 3...$ and $l = $ s, p, d... to denote the total number of number of nodes in the function $= 0, 1, 2...$ and the number of angular nodes $= 0, 1, 2...$ respectively. For example, "2s" denotes a centre–surround function with one radial node and no angular nodes (see figure 1).

For monotonic and non-negative covariance functions, we conjecture that the eigenfunctions of $\mathbf{Q}$ are ordered in eigenvalue by their numbers of nodes such that the eigenfunction $[nl]$ has larger eigenvalue than either $[(n + 1)l]$ or $[n(l + 1)]$. This conjecture is obeyed in all analytical and numerical results we have obtained.

### 2.2    ANALYTIC CALCULATIONS FOR $k_2 = 0$

We have solved analytically for the first three eigenfunctions and eigenvalues of the covariance matrix for layer $\mathcal{B} \to \mathcal{C}$ of Linsker's network, in the continuum limit (Table 1). 1s, the function with no changes of sign, is the principal eigenfunction of $\mathbf{Q}$; 2p, the bilobed oriented function, is the second eigenfunction; and 2s, the centre–surround eigenfunction, is third.[3]

Figure 1(a) shows the first six eigenfunctions for layer $\mathcal{B} \to \mathcal{C}$ of [Linsker, 1986].

**Table 1: The first three eigenfunctions of the operator $Q(r, r')$**
$Q(r, r') = e^{-(r-r')^2/2C}e^{-r'^2/2A}$, where $C$ and $A$ denote the characteristic sizes of the covariance function and synaptic density function. $r$ denotes two-dimensional spatial position relative to the centre of the synaptic arbor, and $r = |r|$. The eigenvalues $\lambda$ are all normalised by the effective number of synapses.

| Name | Eigenfunction | $\lambda/N$ |
|------|---------------|-------------|
| 1s | $e^{-r^2/2R}$ | $lC/A$ |
| 2p | $r\cos\theta e^{-r^2/2R}$ | $l^2C/A$ |
| 2s | $(1 - r^2/r_0^2)e^{-r^2/2R}$ | $l^3C/A$ |

$$R = \frac{C}{2}\left(1 + \sqrt{1 + 4A/C}\right)$$

$$l = \frac{R-C}{R} \quad (0 < l < 1)$$

$$r_0^2 = \frac{2A}{\sqrt{1+4A/C}}$$

**Figure 1: Eigenfunctions of the operator $Q + k_2 J$.**
Largest eigenvalue is in the top row. Eigenvalues (in arbitrary units): (a) $k_2 = 0$: 1s, 2.26; 2p, 1.0; 2s & 3d (only one 3d is shown), 0.41. (b) $k_2 = -3$: 2p, 1.0; 2s, 0.66; 1s, -17.8. The greyscale indicates the range from maximum negative to maximum positive synaptic weight within each eigenfunction. Eigenfunctions of the operator $(e^{-(r-r')^2/2C} + k_2)e^{-r'^2/2A}$ were computed for $C/A = 2/3$ (as used by Linsker for most layer $B \rightarrow C$ simulations) on a circle of radius 12.5 grid intervals, with $\sqrt{A} = 6.15$ grid intervals.

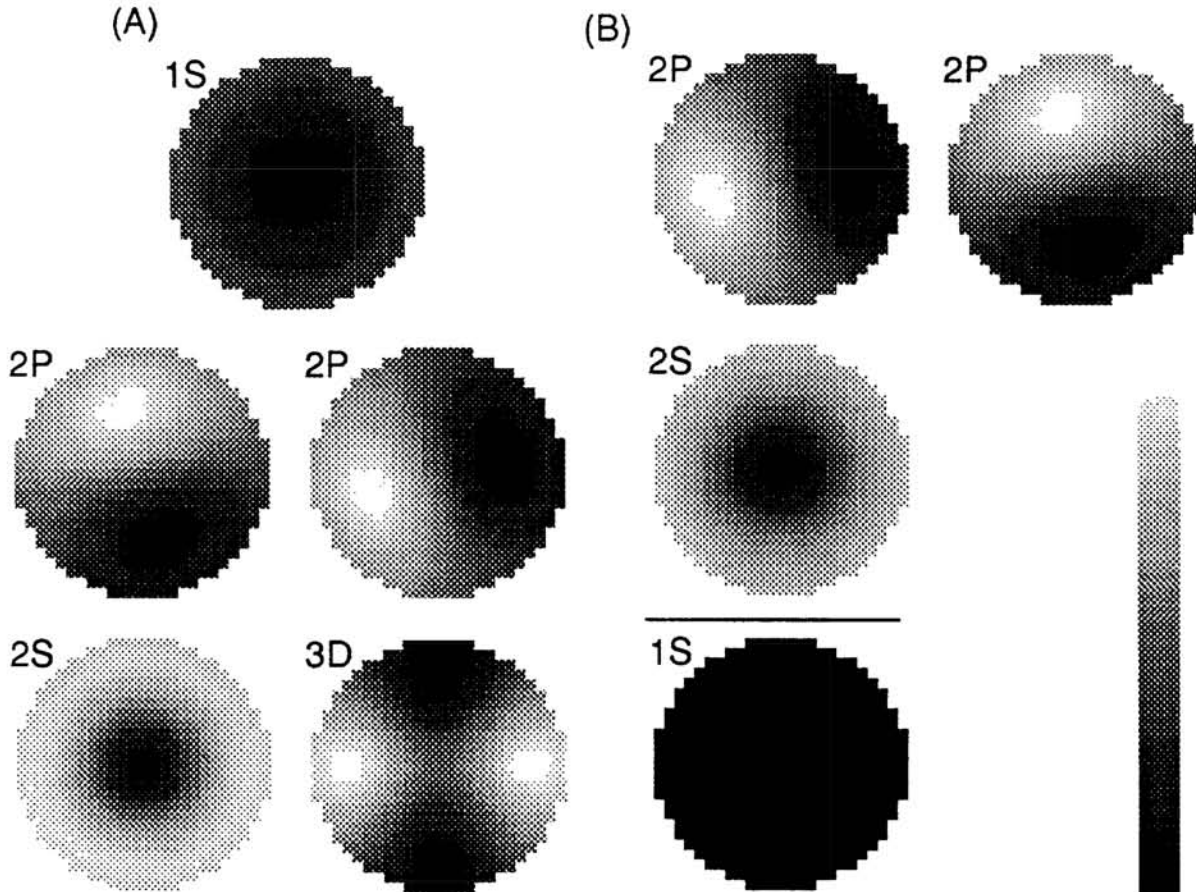

## 3    THE EFFECTS OF THE PARAMETERS $k_1$ AND $k_2$

Varying $k_2$ changes the eigenvectors and eigenvalues of the matrix $\mathbf{Q} + k_2\mathbf{J}$. Varying $k_1$ moves the fixed point of the dynamics with respect to the origin. We now analyse these two changes, and their effects on the dynamics.

**Definition:** Let $\hat{\mathbf{n}}$ be the unit vector in the direction of the DC vector $\mathbf{n}$. We refer to $(\mathbf{w} \cdot \hat{\mathbf{n}})$ as the *DC component* of $\mathbf{w}$. The DC component is proportional to the sum of the synaptic strengths in a weight vector. For example, 2p and all the other eigenfunctions with angular nodes have zero DC component. Only the s–modes have a non–zero DC component.

### 3.1    GENERAL THEOREM: THE EFFECT OF $k_2$

We now characterise the effect of adding $k_2\mathbf{J}$ to *any* covariance matrix $\mathbf{Q}$.

**Theorem 1** *For any covariance matrix $\mathbf{Q}$, the spectrum of eigenvectors and eigenvalues of $\mathbf{Q} + k_2\mathbf{J}$ obeys the following:*
*1. Eigenvectors of $\mathbf{Q}$ with no DC component, and their eigenvalues, are unaffected by $k_2$.*
*2. The other eigenvectors, with non–zero DC component, vary with $k_2$. Their eigenvalues increase continuously and monotonically with $k_2$ between asymptotic limits such that the upper limit of one eigenvalue is the lower limit of the eigenvalue above.*
*3. There is at most one negative eigenvalue.*
*4. All but one of the eigenvalues remain finite. In the limits $k_2 \to \pm\infty$ there is a DC eigenvector $\hat{\mathbf{n}}$ with eigenvalue $\to k_2N$, where $N$ is the dimensionality of $\mathbf{Q}$, i.e. the number of synapses.*

The properties stated in this theorem, whose proof is in [MacKay, Miller, 1990], are summarised pictorially by the spectral structure shown in figure 2.

### 3.2    IMPLICATIONS FOR LINSKER'S SYSTEM

For Linsker's circularly symmetric systems, all the eigenfunctions with angular nodes have zero DC component and are thus independent of $k_2$. The eigenvalues that vary with $k_2$ are those of the s–modes. The leading s–modes at $k_2 = 0$ are 1s, 2s; as $k_2$ is decreased to $-\infty$, these modes transform continuously into 2s, 3s respectively (figure 2).[4] 1s becomes an eigenvector with negative eigenvalue, and it approaches the DC vector $\hat{\mathbf{n}}$. This eigenvector enforces a constraint $\mathbf{w} \cdot \hat{\mathbf{n}} = \mathbf{w}^{\text{FP}} \cdot \hat{\mathbf{n}}$, and thus determines that the final average synaptic strength is equal to $\mathbf{w}^{\text{FP}} \cdot \mathbf{n}/N$.

Linsker used $k_2 = -3$ in [Linsker, 1986]. This value of $k_2$ is sufficiently large that the properties of the $k_2 \to -\infty$ limit hold [MacKay, Miller, 1990], and in the following we concentrate interchangeably on $k_2 = -3$ and $k_2 \to -\infty$. The computed eigenfunctions for Linsker's system at layer $\mathcal{B} \to \mathcal{C}$ are shown in figure 1(b) for

**Figure 2: General spectrum of eigenvalues of $Q + k_2 J$ as a function of $k_2$.**
A: Eigenvectors with DC component. B: Eigenvectors with zero DC component.
C: Adjacent DC eigenvalues share a common asymptote. D: There is only one
negative eigenvalue.
The annotations in brackets refer to the eigenvectors of Linsker's system.

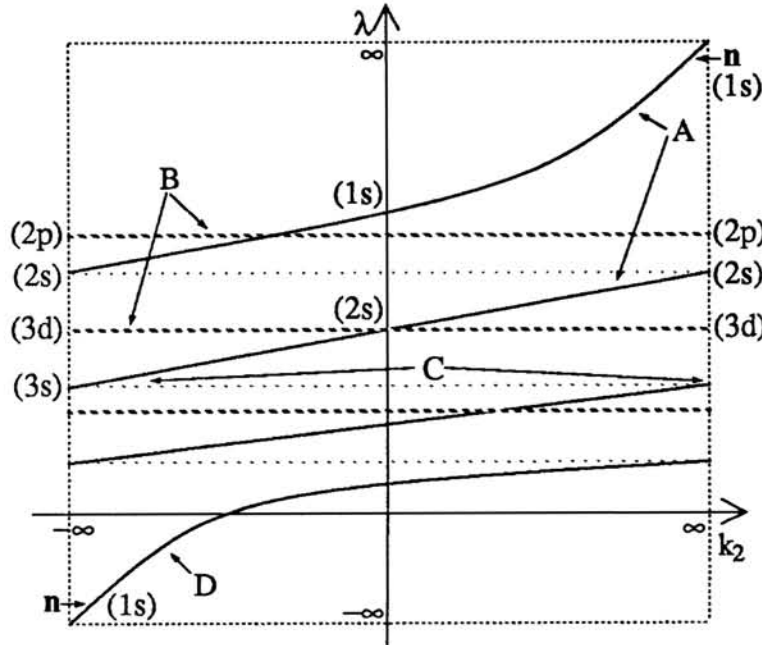

$k_2 = -3$. The principal eigenfunction is 2p. The centre–surround eigenfunction 2s
is the principal *symmetric* eigenfunction, but it still has smaller eigenvalue than 2p.

## 3.3  EFFECT OF $k_1$

Varying $k_1$ changes the location of the fixed point of equation (2). From equation
(3), the fixed point is displaced from the origin only in the direction of eigenvectors
that have non–zero DC component, that is, only in the direction of the s–modes.
This has two important effects, as discussed in section 1: a) The s–modes are given
a head start in growth rate that increases as $k_1$ is increased. In particular, the
principal s–mode, the centre–surround eigenvector 2s, may outgrow the principal
eigenvector 2p. b) The constraint surface is moved when $k_1$ is changed. For large
negative $k_2$, the constraint surface fixes the average synaptic strength in the final
weight vector. To leading order in $1/k_2$, Linsker showed that the constraint is:
$\sum w_j = k_1/|k_2|$.[5]

## 3.4  SUMMARY OF THE EFFECTS OF $k_1$ AND $k_2$

We can now anticipate the explanation for the emergence of centre–surround cells:
For $k_1 = 0$, $k_2 = 0$, the dynamics are dominated by 1s. The centre–surround

covariance (averaged over $i$ and $j$). The additional term largely resolves the discrepancy between
Linsker's $g$ and $k_1/|k_2|$ in [Linsker, 1986].

eigenfunction 2s is third in line behind 2p, the bi–lobed function. Making $k_2$ large and negative removes 1s from the lead. 2p becomes the principal eigenfunction and dominates the dynamics for $k_1 \simeq 0$, so that the circular symmetry is broken. Finally, increasing $k_1/|k_2|$ gives a head start to the centre–surround function 2s. Increasing $k_1/|k_2|$ also increases the final average synaptic strength, so large $k_1/|k_2|$ also produces a large DC bias. The centre–surround regime therefore lies sandwiched between a 2p–dominated regime and an all–excitatory regime. $k_1/|k_2|$ has to be large enough that 2s dominates over 2p, and small enough that the DC bias does not obscure the centre–surround structure. We estimate this parameter regime in [MacKay, Miller, 1990], and show that the boundary between the 2s– and 2p–dominated regimes found by simulated annealing on the energy function may be different from the boundary found by simulating the time–development of equation (1), which depends on the initial conditions.

## 4   CONCLUSIONS AND DISCUSSION

For Linsker's $\mathcal{B} \to \mathcal{C}$ connections, we predict four main parameter regimes for varying $k_1$ and $k_2$.[6] These regimes, shown in figure 3, are dominated by the following weight structures:

| | |
|---|---|
| $k_2 = 0$, $k_1 = 0$: | The principal eigenvector of $\mathbf{Q}$, 1s. |
| $k_2 =$ large positive and/or $k_1 =$ large | The flat DC weight vector, which leads to the same saturated structures as 1s. |
| $k_2 =$ large negative, $k_1 \simeq 0$ | The principal eigenvector of $\mathbf{Q} + k_2\mathbf{J}$ for $k_2 \to -\infty$, 2p. |
| $k_2 =$ large negative, $k_1 =$ intermediate | The principal circularly symmetric function which is given a head start, 2s. |

Higher layers of Linsker's network can be analysed in terms of the same four regimes; the principal eigenvectors are altered, so that different structures can emerge. The development of the interesting cells in Linsker's system depends on the use of negative synapses and on the use of the terms $k_1$ and $k_2$ to enforce a constraint on the final percentages of positive and negative synapses. Both of these may be biologically problematic [Miller, 1990]. Linsker suggested that the emergence of centre–surround structures may depend on the peaked synaptic density function that he used [Linsker, 1986, page 7512]. However, with a flat density function, the eigenfunctions are qualitatively unchanged, and centre–surround structures can emerge by the same mechanism.

### Acknowledgements

D.J.C.M. is supported by a Caltech Fellowship and a Studentship from SERC, UK.

K.D.M. thanks M. P. Stryker for encouragement and financial support while this work was undertaken. K.D.M. was supported by an N.E.I. Fellowship and the In-

**Figure 3: Parameter regimes for Linsker's system.** The DC bias is approximately constant along the radial lines, so each of the regimes with large negative $k_2$ is wedge-shaped.

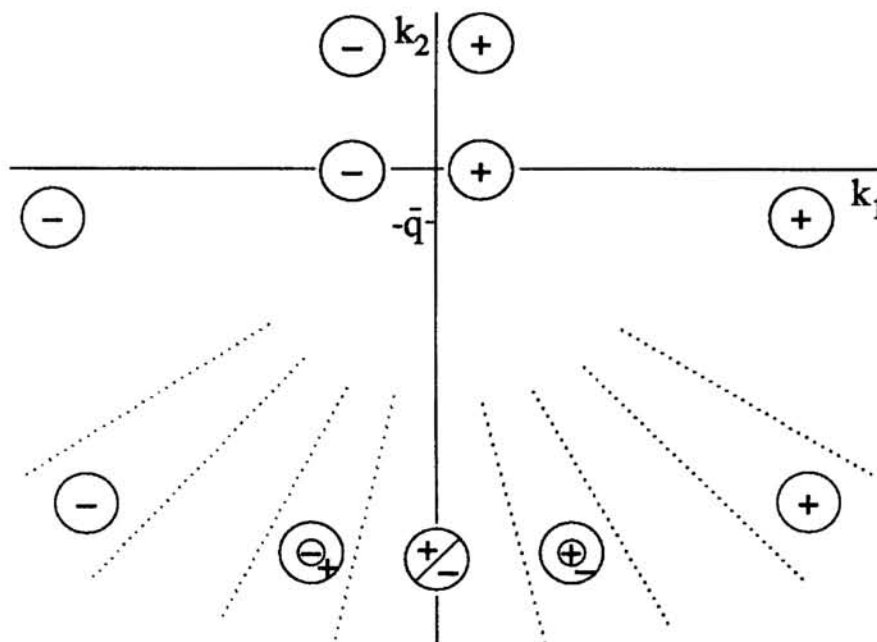

ternational Joint Research Project Bioscience Grant to M. P. Stryker (T. Tsumoto, Coordinator) from the N.E.D.O., Japan.

This collaboration would have been impossible without the internet/NSFnet, long may their daemons flourish.

## Footnotes

[1] Our definition of equation 1 differs from Linsker's by the omission of a factor of $1/N$ before the sum term, where $N$ is the number of synapses.

[2] The indices $a$ and $b$ will be used to denote the eigenvector basis for $\mathbf{w}$, while the indices $i$ and $j$ will be used for the synaptic basis.

[3]2s is degenerate with 3d at $k_2 = 0$.

[4] The 2s eigenfunctions at $k_2 = 0$ and $k_2 = -\infty$ both have one radial node, but are not identical functions.

[5] To second order, this expression becomes $\sum w_j = k_1/|k_2 + \bar{q}|$, where $\bar{q} = \langle Q_{ij} \rangle$, the average

[6]not counting the symmetric regimes $(k_1, k_2) \leftrightarrow (-k_1, k_2)$ in which all the weight structures are inverted in sign.

# References

[Linsker, 1986] R. Linsker. From Basic Network Principles to Neural Architecture (series), *PNAS USA*, **83**, Oct.-Nov. 1986, pp. 7508-7512, 8390-8394, 8779-8783.

[Linsker, 1988] R. Linsker. Self-Organization in a Perceptual Network, *Computer*, March 1988.

[Miller, 1990] K.D. Miller. "Correlation-based mechanisms of neural development," in *Neuroscience and Connectionist Theory*, M.A. Gluck and D.E. Rumelhart, Eds. (Lawrence Erlbaum Associates, Hillsboro NJ) (in press).

[MacKay, Miller, 1990] D.J.C. MacKay and K.D. Miller. "Analysis of Linsker's Simulations of Hebbian rules" (submitted to *Neural Computation*); and "Analysis of Linsker's application of Hebbian rules to linear networks" (submitted to *Network*).